# Shooting Craps in Search of an Optimal Strategy for Training Connectionist Pattern Classifiers

**J. B. Hampshire II** and **B. V. K. Vijaya Kumar**
Department of Electrical & Computer Engineering
Carnegie Mellon University
Pittsburgh, PA 15213-3890
hamps@speech1.cs.cmu.edu and kumar@gauss.ece.cmu.edu

## Abstract

We compare two strategies for training connectionist (as well as non-connectionist) models for statistical pattern recognition. The *probabilistic strategy* is based on the notion that Bayesian discrimination (i.e., optimal classification) is achieved when the classifier learns the *a posteriori* class distributions of the random feature vector. The *differential strategy* is based on the notion that the identity of the largest class *a posteriori* probability of the feature vector is all that is needed to achieve Bayesian discrimination. Each strategy is directly linked to a family of objective functions that can be used in the supervised training procedure. We prove that the probabilistic strategy — linked with error measure objective functions such as mean-squared-error and cross-entropy — typically used to train classifiers necessarily requires larger training sets and more complex classifier architectures than those needed to approximate the Bayesian discriminant function. In contrast, we prove that the differential strategy — linked with *classification figure-of-merit* objective functions ($CFM_{mono}$) [3] — requires the minimum classifier functional complexity and the fewest training examples necessary to approximate the Bayesian discriminant function with specified precision (measured in probability of error). We present our proofs in the context of a game of chance in which an unfair $C$-sided die is tossed repeatedly. We show that this rigged game of dice is a paradigm at the root of all statistical pattern recognition tasks, and demonstrate how a simple extension of the concept leads us to a general information-theoretic model of sample complexity for statistical pattern recognition.

# 1    Introduction

Creating a connectionist pattern classifier that generalizes well to novel test data has recently focussed on the process of finding the network architecture with the minimum functional complexity necessary to model the training data accurately (see, for example, the works of Baum, Cover, Haussler, and Vapnik). Meanwhile, relatively little attention has been paid to the effect on generalization of the objective function used to train the classifier. In fact, the choice of objective function used to train the classifier is tantamount to a choice of training strategy, as described in the abstract [2, 3].

We formulate the proofs outlined in the abstract in the context of a rigged game of dice in which an unfair $C$-sided die is tossed repeatedly. Each face of the die has some probability of turning up. We assume that one face is always more likely than all the others. As a result, all the probabilities may be different, but at most $C - 1$ of them can be identical. The objective of the game is to identify the most likely die face with specified high confidence. The relationship between this rigged dice paradigm and statistical pattern recognition becomes clear if one realizes that *a single unfair die is analogous to a specific point on the domain of the random feature vector being classified.* Just as there are specific class probabilities associated with each *point* in feature vector space, each *die* has specific probabilities associated with each of its faces. The number of faces on the die equals the number of classes associated with the analogous point in feature vector space. Identifying the most likely die face is equivalent to identifying the maximum class *a posteriori* probability for the analogous point in feature vector space — the requirement for Bayesian discrimination. We formulate our proofs for the case of a single die, and conclude by showing how a simple extension of the mathematics leads to general expressions for pattern recognition involving both discrete and continuous random feature vectors.

**Authors' Note:** In the interest of brevity, our proofs are posed as answers to questions that pertain to the rigged game of dice. It is hoped that the reader will find the relevance of each question/answer to statistical pattern recognition clear. Owing to page limitations, we cannot provide our proofs in full detail; the reader seeking such detail should refer to [1]. Definitions of symbols used in the following proofs are given in table 1.

## 1.1    A Fixed-Point Representation

The $M_q$-bit approximation $q_M[x]$ to the real number $x \in (-1, 1]$ is of the form

$$MSB \quad \text{(most significant bit)} \quad = \quad \text{sign}[x]$$

$$MSB - 1 \quad = \quad 2^{-1} \tag{1}$$
$$\downarrow$$
$$LSB \quad \text{(least significant bit)} \quad = \quad 2^{-(M_q-1)}$$

with the specific value defined as the mid-point of the $2^{-(M_q-1)}$-wide interval in which $x$ is located:

$$q_M[x] \triangleq \begin{cases} \text{sign}[x] \cdot \left( \lfloor |x| \cdot 2^{(M_q-1)} \rfloor \cdot 2^{-(M_q-1)} + 2^{-M_q} \right), & |x| < 1 \\ \text{sign}[x] \cdot \left( 1 - 2^{-M_q} \right), & |x| = 1 \end{cases} \tag{2}$$

The lower and upper bounds on the quantization interval are

$$L_{M_q}[x] < x \leq U_{M_q}[x] \tag{3}$$

Table 1: Definitions of symbols used to describe die faces, probabilities, probabilistic differences, and associated estimates.

| Symbol | Definition |
| --- | --- |
| $\omega_{rj}$ | The true $j$th most likely die face ($\widehat{\omega_{rj}}$ is the estimated $j$th most likely face). |
| $P(\omega_{rj})$ | The probability of the true $j$th most likely die face. |
| $k_{rj}$ | The number of occurrences of the true $j$th most likely die face. |
| $\hat{P}(\omega_{rj})$ | An empirical estimate of the probability of the true $j$th most likely die face: $\hat{P}(\omega_{rj}) = \frac{k_{rj}}{n}$ (note $n$ denotes the sample size) |
| $\Delta_{ri}$ | The probabilistic difference involving the true rankings and probabilities of the $C$ die faces: $\Delta_{ri} = P(\omega_{ri}) - \sup_{j \neq i} P(\omega_{rj})$ |
| $\hat{\Delta}_{ri}$ | The probabilistic difference involving the true rankings but *empirically estimated* probabilities of the $C$ die faces: $\hat{\Delta}_{ri} = \hat{P}(\omega_{ri}) - \sup_{j \neq i} \hat{P}(\omega_{rj}) = \frac{k_{ri} - \sup_{j \neq i} k_{rj}}{n}$ |

where

$$L_{M_q}[x] = q_M[x] - 2^{-M_q} \tag{4}$$

and

$$U_{M_q}[x] = q_M[x] + 2^{-M_q} \tag{5}$$

The fixed-point representation described by (1) – (5) differs from standard fixed-point representations in its choice of quantization interval. The choice of (2) – (5) represents zero as a negative — more precisely, a *non-positive* — finite precision number. See [1] for the motivation of this format choice.

## 1.2  A Mathematical Comparison of the Probabilistic and Differential Strategies

The probabilistic strategy for identifying the most likely face on a die with $C$ faces involves estimating the $C$ face probabilities. In order for us to distinguish $\hat{P}(\omega_{r1})$ from $\hat{P}(\omega_{r2})$, we must choose $M_q$ (i.e. the number of bits in our fixed-point representation of the estimated probabilities) such that

$$q_M[\hat{P}(\omega_{r1})] > q_M[\hat{P}(\omega_{r2})] \tag{6}$$

The distinction between the differential and probabilistic strategies is made more clear if one considers the way in which the $M_q$-bit approximation $\hat{\Delta}_{r1}$ is computed from a random sample containing $k_{r1}$ occurrences of die face $\hat{\omega}_{r1}$ and $k_{r2}$ occurrences of die face $\hat{\omega}_{r2}$. For the differential strategy

$$\hat{\Delta}_{r1 \text{ differential}} \equiv q_M \left[ \frac{k_{r1} - k_{r2}}{n} \right] \tag{7}$$

and for the probabilistic strategy

$$\hat{\Delta}_{r1 \text{ probabilistic}} \equiv q_M \left[ \frac{k_{r1}}{n} \right] - q_M \left[ \frac{k_{r2}}{n} \right] \tag{8}$$

where

$$\Delta_i \overset{\triangle}{=} P(\omega_i) - \sup_{j \neq i} P(\omega_j) \quad i = 1, 2, \ldots C \tag{9}$$

Note that when $i = r1$

$$\Delta_{i=r1} = P(\omega_{r1}) - P(\omega_{r2}) \tag{10}$$

and when $i \neq r1$

$$\Delta_{i \neq r1} = P(\omega_i) - P(\omega_{r1}) \tag{11}$$

Note also

$$\Delta_{r1} = -\Delta_{r2} \tag{12}$$

Since

$$\sum_{i=1}^{C} \Delta_i = \sum_{i=r3}^{rC} P(\omega_i) - (C - 2) P(\omega_{r1}) \tag{13}$$

we can show that the $C$ differences of (9) yield the $C$ probabilities by

$$P(\omega_{r1}) = \frac{1}{C} \left[ 1 - \sum_{i=r2}^{rC} \Delta_i \right] \tag{14}$$

$$P(\omega_{rj}) = \Delta_{rj} + P(\omega_{r1}) \quad \forall j > 1$$

Thus, estimating the $C$ differences of (9) is equivalent to estimating the $C$ probabilities $P(\omega_1), P(\omega_2), \ldots, P(\omega_C)$.

Clearly, the sign of $\hat{\Delta}_{r1}$ in (7) is modeled correctly (i.e., $\hat{\Delta}_{r1\,differential}$ can correctly identify the most likely face) when $M_q = 1$, while this is typically not the case for $\hat{\Delta}_{r1\,probabilistic}$ in (8). In the latter case, $\hat{\Delta}_{r1\,probabilistic}$ is zero when $M_q = 1$ because $q_m[\hat{P}(\omega_{r1})]$ and $q_M[\hat{P}(\omega_{r2})]$ are indistinguishable for $M_q$ below some minimal value implied by (6). That minimal value of $M_q$ can be found by recognizing that the number of bits necessary for (6) to hold for asymptotically large $n$ (i.e., for the quantized difference in (8) to exceed one $LSB$) is

$$M_{q\,min} = \begin{cases} \underbrace{1}_{\text{sign bit}} + \underbrace{\lceil -\log_2 \lceil \Delta_{r1} \rceil \rceil}_{\text{magnitude bits}}, & -\log_2 [P(\omega_{rj})] \ni \mathcal{Z}^+ \quad j \in \{1,2\} \\ \\ \underbrace{1}_{\text{sign bit}} + \underbrace{\lceil -\log_2 \lceil \Delta_{r1} \rceil \rceil + 1}_{\text{magnitude bits}}, & \text{otherwise} \end{cases}$$
$$\tag{15}$$

where $\mathcal{Z}^+$ represents the set of all positive integers. Note that the conditional nature of $M_{q\,min}$ in (15) prevents the case in which $\lim_{\varepsilon \to 0} P(\omega_{r1}) - \varepsilon = L_{M_q}[P(\omega_{r1})]$ or $P(\omega_{r2}) = U_{M_q}[P(\omega_{r2})]$; either case would require an infinitely large sample size before the variance of the corresponding estimated probability became small enough to distinguish $q_M[\hat{P}(\omega_{r1})]$ from $q_M[\hat{P}(\omega_{r2})]$. The sign bit in (15) is not required to estimate the probabilities themselves in (8), but it is necessary to compute the difference between the two probabilities in that equation — this difference being the ultimate computation by which we choose the most likely die face.

## 1.3   The Sample Complexity Product

We introduce the *sample complexity product* (SCP) as a measure of both the number of samples and the functional complexity (measured in bits) required to identify the most likely face of an unfair die with specified probability.

$$\text{SCP} \stackrel{\triangle}{=} n \cdot M_q \quad s.t. \quad P(\text{most likely face correctly ID'd}) \geq \alpha \quad (16)$$

# 2   A Comparison of the Sample Complexity Requirements for the Probabilistic and Differential Strategies

**Axiom 1** *We view the number of bits $M_q$ in the finite-precision approximation $q_M[x]$ to the real number $x \in (-1, 1]$ as a measure of the approximation's functional complexity. That is, the functional complexity of an approximation is the number of bits with which it represents a real number on $(-1, 1]$.*

**Assumption 1** *If $\hat{P}(\omega_{r1}) > \hat{P}(\omega_{r2})$, then $\hat{P}(\omega_{r1})$ will be greater than $\hat{P}(\omega_{rj})$ $\forall j > 2$ (see [1] for an analysis of cases in which this assumption is invalid).*

**Question:** What is the probability that the most likely face of an unfair die will be empirically identifiable after $n$ tosses?

**Answer for the probabilistic strategy:**

$$P\left(q_M[\hat{P}(\omega_{r1})] > q_M[\hat{P}(\omega_{rj})], \ \forall j > 1\right)$$

$$\cong \ n! \sum_{k_{r1}=\lambda_1}^{\upsilon_1} \frac{P(\omega_{r1})^{k_{r1}}}{k_{r1}!} \left[ \sum_{k_{r2}=\lambda_2}^{\upsilon_2} \frac{P(\omega_{r2})^{k_{r2}} (1 - P(\omega_{r1}) - P(\omega_{r2}))^{(n-k_{r1}-k_{r2})}}{k_{r2}! \, (n - k_{r1} - k_{r2})!} \right] \quad (17)$$

where

$$\begin{aligned}
\lambda_1 &= \max\left( \mathcal{B} + 1, \frac{n - k_{r2}}{C - 1} + 1 \right) \qquad \forall C > 2 \\
\upsilon_1 &= n \\
\lambda_2 &= 0 \qquad\qquad\qquad\qquad\qquad\qquad\qquad\qquad\quad (18) \\
\upsilon_2 &= \min(\mathcal{B}, n - k_{r1}) \\
\mathcal{B} &= \{\mathcal{B}_{M_q}\} = k_{U_{M_q}}[P(\omega_{r2})] = k_{L_{M_q}}[P(\omega_{r1})] - 1
\end{aligned}$$

There is a simple recursion in [1] by which every possible boundary for $M_q$-bit quantization leads to itself and two additional boundaries in the set $\{\mathcal{B}_{M_q}\}$ for $(M_q + 1)$-bit quantization.

**Answer for the differential strategy:**

$$P\left(L_{M_q}[\Delta_{r1}] < \hat{\Delta}_{r1} \leq U_{M_q}[\Delta_{r1}], \ \hat{\Delta}_{rj} < 0 \ \forall j > 1\right)$$

$$\cong \; n! \sum_{k_{r1}=\lambda_1}^{\upsilon_1} \frac{P(\omega_{r1})^{k_{r1}}}{k_{r1}!} \left[ \sum_{k_{r2}=\lambda_2}^{\upsilon_2} \frac{P(\omega_{r2})^{k_{r2}} (1 - P(\omega_{r1}) - P(\omega_{r2}))^{(n-k_{r1}-k_{r2})}}{k_{r2}! \, (n - k_{r1} - k_{r2})!} \right] \quad (19)$$

where

$$
\begin{aligned}
\lambda_1 &= \max\left( k_{L_{M_q}}[\Delta_{r1}], \; \frac{n - k_{r2}}{C - 1} + 1 \right) && \forall \, C > 2 \\
\upsilon_1 &= n && (20) \\
\lambda_2 &= \max\left( 0, \; k_{r1} - k_{U_{M_q}}[\Delta_{r1}] \right) \\
\upsilon_2 &= \min\left( k_{r1} - k_{L_{M_q}}[\Delta_{r1}], \; n - k_{r1} \right)
\end{aligned}
$$

Since the multinomial distribution is positive semi-definite, it should be clear from a comparison of (17) – (18) and (19) – (20) that $P\left( L_{M_q}[\Delta_{r1}] < \hat{\Delta}_{r1} \leq U_{M_q}[\Delta_{r1}] \right)$ is largest (and larger than any possible $P\left( q_M[\hat{P}(\omega_{r1})] > q_M[\hat{P}(\omega_{rj})], \; \forall j > 1 \right)$) for a given sample size $n$ when the differential strategy is employed with $M_q = 1$ such that $L_{M_q}[\Delta_{r1}] = 0$ and $U_{M_q}[\Delta_{r1}] = 1$ (i.e., $k_{L_{M_q}}[\Delta_{r1}] = 1$ and $k_{U_{M_q}}[\Delta_{r1}] = n$). The converse is also true, to wit:

**Theorem 1** *For a fixed value of n in (19), the 1-bit approximation to $\Delta_{r1}$ yields the highest probability of identifying the most likely die face $\omega_{r1}$.*

It can be shown that theorem 1 does not depend on the validity of assumption 1 [1]. Given Axiom 1, the following corollary to theorem 1 holds:

**Corollary 1** *The differential strategy's minimum-complexity 1-bit approximation of $\Delta_{r1}$ yields the highest probability of identifying the most likely die face $\omega_{r1}$ for a given number of tosses n.*

**Corollary 2** *The differential strategy's minimum-complexity 1-bit approximation of $\Delta_{r1}$ requires the smallest sample size necessary ($n_{min}$) to identify $P(\omega_{r1})$ — and thereby the most likely die face $\omega_{r1}$ — correctly with specified confidence. Thus, the differential strategy requires the minimum SCP necessary to identify the most likely die face with specified confidence.*

## 2.1   Theoretical Predictions versus Empirical Results

Figures 1 and 2 compare theoretical predictions of the number of samples $n$ and the number of bits $M_q$ necessary to identify the most likely face of a particular die versus the actual requirements obtained from 1000 games (3000 tosses of the die in each game). The die has five faces with probabilities $P(\omega_{r1}) = 0.37$, $P(\omega_{r2}) = 0.28$, $P(\omega_{r3}) = 0.2$, $P(\omega_{r4}) = 0.1$, and $P(\omega_{r1}) = 0.05$. The theoretical predictions for $M_q$ and $n$ (arrows with boxed labels based on iterative searches employing equations (17) and (19)) that would with 0.95 confidence correctly identify the most likely die face $\omega_{r1}$ are shown to correspond with the empirical results: in figure 1 the empirical 0.95 confidence interval is marked by the lower bound of the dark gray and the upper bound of the light gray; in figure 2 the empirical 0.95 confidence interval is marked by the lower bound of the $\hat{P}(\omega_{r1})$ distribution and the upper bound of the

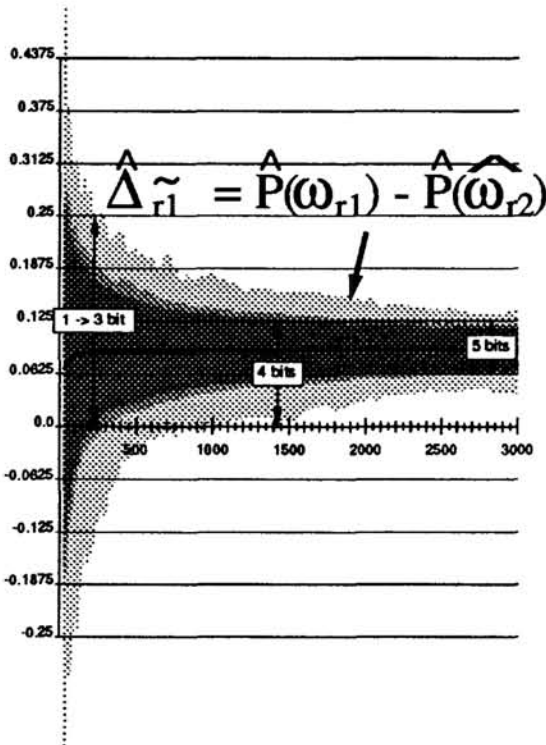

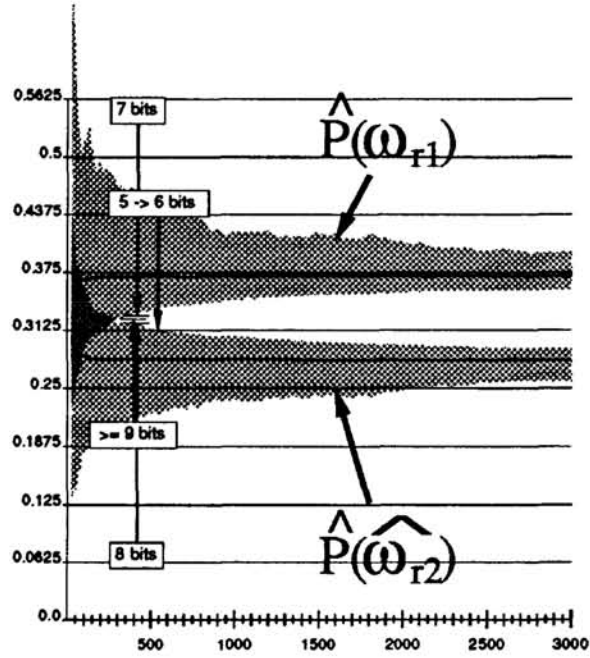

Figure 1: Theoretical predictions of the number of tosses needed to identify the most likely face $\omega_{r1}$ with 95% confidence (Die 1): Differential strategy prediction superimposed on empirical results of 1000 games (3000 tosses each).

Figure 2: Theoretical predictions of the number of tosses needed to identify the most likely face $\omega_{r1}$ with 95% confidence (Die 1): Probabilistic strategy prediction superimposed on empirical results of 1000 games (3000 tosses each).

$\hat{P}(\omega_{r2})$ distribution. These figures illustrate that the differential strategy's minimum SCP is 227 ($n = 227$, $M_q = 1$) while the minimum SCP for the probabilistic strategy is 2720 ($n = 544$, $M_q = 5$). A complete tabulation of SCP as a function of $P(\omega_{r1})$, $P(\omega_{r2})$, and the worst-case choice for $C$ (the number of classes/die faces) is given in [1].

## 3  Conclusion

The sample complexity product (SCP) notion of functional complexity set forth herein is closely aligned with the complexity measures of Kolmogorov and Rissanen [4, 6]. We have used it to prove that the differential strategy for learning the Bayesian discriminant function is optimal in terms of its minimum requirements for classifier functional complexity and number of training examples when the classification task is identifying the most likely face of an unfair die. It is relatively straightforward to extend theorem 1 and its corollaries to the general pattern recognition case in order to show that the expected SCP for the 1-bit differential strategy

$$E\,[SCP]_{differential} \cong \int_{\mathbf{X}} n_{min}\left[P\left(\omega_{r1} \mid \mathbf{x}\right), P\left(\omega_{r2} \mid \mathbf{x}\right)\right] \cdot \underbrace{M_{q\,min}\left[P\left(\omega_{r1} \mid \mathbf{x}\right), P\left(\omega_{r2} \mid \mathbf{x}\right)\right]}_{=1} \rho(\mathbf{x})d\mathbf{x}$$

(21)

(or the discrete random vector analog of this equation) is minimal [1]. This is because $n_{min}$ is by corollary 2 the smallest sample size necessary to distinguish any and all $P(\omega_{r1})$ from

lesser $P(\omega_{r2})$. The resulting analysis confirms that the classifier trained with the differential strategy for statistical pattern recognition (i.e., using a $CFM_{mono}$ objective function) has the highest probability of learning the Bayesian discriminant function when the functional capacity of the classifier and the available training data are both limited.

The relevance of this work to the process of designing and training robust connectionist pattern classifiers is evident if one considers the practical meaning of the terms $n_{min}\left[P\left(\omega_{r1}\mid\mathbf{x}\right), P\left(\omega_{r2}\mid\mathbf{x}\right)\right]$ and $M_{q\,min}\left[P\left(\omega_{r1}\mid\mathbf{x}\right), P\left(\omega_{r2}\mid\mathbf{x}\right)\right]$ in the sample complexity product of (21). Given one's choice of connectionist model to employ as a classifier, the $M_{q\,min}$ term dictates the minimum necessary connectivity of that model. For example, (21) can be used to prove that a partially connected radial basis function (RBF) with trainable variance parameters and three hidden layer "nodes" has the minimum $M_q$ necessary for Bayesian discrimination in the 3-class task described by [5]. However, because *both* SCP terms are functions of the probabilistic nature of the random feature vector being classified *and the learning strategy employed*, that minimal RBF architecture *will only yield Bayesian discrimination if trained using the differential strategy*. The probabilistic strategy requires significantly more functional complexity in the RBF in order to meet the requirements of the probabilistic strategy's SCP [1]. Philosophical arguments regarding the use of the differential strategy in lieu of the more traditional probabilistic strategy are discussed at length in [1].

### Acknowledgement

This research was funded by the Air Force Office of Scientific Research under grant AFOSR-89-0551. We gratefully acknowledge their support.

### References

[1] J. B. Hampshire II. *A Differential Theory of Statistical Pattern Recognition.* PhD thesis, Carnegie Mellon University, Department of Electrical & Computer Engineering, Hammerschlag Hall, Pittsburgh, PA 15213-3890, 1992. manuscript in progress.

[2] J. B. Hampshire II and B. A. Pearlmutter. Equivalence Proofs for Multi-Layer Perceptron Classifiers and the Bayesian Discriminant Function. In Touretzky, Elman, Sejnowski, and Hinton, editors, *Proceedings of the 1990 Connectionist Models Summer School*, pages 159–172, San Mateo, CA, 1991. Morgan-Kaufmann.

[3] J. B. Hampshire II and A. H. Waibel. A Novel Objective Function for Improved Phoneme Recognition Using Time-Delay Neural Networks. *IEEE Transactions on Neural Networks*, 1(2):216–228, June 1990. A revised and extended version of work first presented at the 1989 International Joint Conference on Neural Networks, vol. I, pp. 235-241.

[4] A. N. Kolmogorov. Three Approaches to the Quantitative Definition of Information. *Problems of Information Transmission*, 1(1):1–7, Jan. - Mar. 1965. Faraday Press translation of Problemy Peredachi Informatsii.

[5] M. D. Richard and R. P. Lippmann. Neural Network Classifiers Estimate Bayesian *a posteriori* Probabilities. *Neural Computation*, 3(4):461–483, 1991.

[6] J. Rissanen. Modeling by shortest data description. *Automatica*, 14:465–471, 1978.